# An Annealed Self-Organizing Map for Source Channel Coding

**Matthias Burger, Thore Graepel, and Klaus Obermayer**
Department of Computer Science
Technical University of Berlin
FR 2-1, Franklinstr. 28/29, 10587 Berlin, Germany
{burger, graepel2, oby}@cs.tu-berlin.de

## Abstract

We derive and analyse robust optimization schemes for noisy vector quantization on the basis of deterministic annealing. Starting from a cost function for central clustering that incorporates distortions from channel noise we develop a soft topographic vector quantization algorithm (STVQ) which is based on the maximum entropy principle and which performs a maximum-likelihood estimate in an expectation-maximization (EM) fashion. Annealing in the temperature parameter $\beta$ leads to phase transitions in the existing code vector representation during the cooling process for which we calculate critical temperatures and modes as a function of eigenvectors and eigenvalues of the covariance matrix of the data and the transition matrix of the channel noise. A whole family of vector quantization algorithms is derived from STVQ, among them a deterministic annealing scheme for Kohonen's self-organizing map (SOM). This algorithm, which we call SSOM, is then applied to vector quantization of image data to be sent via a noisy binary symmetric channel. The algorithm's performance is compared to those of LBG and STVQ. While it is naturally superior to LBG, which does not take into account channel noise, its results compare very well to those of STVQ, which is computationally much more demanding.

## 1   INTRODUCTION

Noisy vector quantization is an important lossy coding scheme for data to be transmitted over noisy communication lines. It is especially suited for speech and image data which in many applieations have to be transmitted under low bandwidth / high noise level conditions. Following the idea of (Farvardin, 1990) and (Luttrell, 1989) of jointly optimizing the codebook and the data representation w.r.t. to a given channel noise we apply a deterministic annealing scheme (Rose, 1990; Buhmann, 1997) to the problem and develop a

soft topographic vector quantization algorithm (STVQ) (cf. Heskes, 1995; Miller, 1994). From STVQ we can derive a class of vector quantization algorithms, among which we find SSOM, a deterministic annealing variant of Kohonen's self-organizing map (Kohonen, 1995), as an approximation. While the SSOM like the SOM does not minimize any known energy function (Luttrell, 1989) it is computationally less demanding than STVQ. The deterministic annealing scheme enables us to use the neighborhood function of the SOM solely to encode the desired transition probabilities of the channel noise and thus opens up new possibilities for the usage of SOMs with arbitrary neighborhood functions. We analyse phase transitions during the annealing and demonstrate the performance of SSOM by applying it to lossy image data compression for transmission via noisy channels.

## 2  DERIVATION OF A CLASS OF VECTOR QUANTIZERS

Vector quantization is a method of encoding data by grouping the data vectors and providing a representative in data space for each group. Given a set $\mathcal{X}$ of data vectors $x_i \in \Re^d$, $i = 1, \ldots, D$, the objective of vector quantization is to find a set $\mathcal{W}$ of code vectors $w_r$, $r = 0, \ldots, N-1$, and a set $\mathcal{M}$ of binary assignment variables $m_{ir}$, $\sum_r m_{ir} = 1$, $\forall i$, such that a given cost function

$$E(\mathcal{M}, \mathcal{W} \mid \mathcal{X}) = \sum_i \sum_r m_{ir} E_r(x_i, \mathcal{W}) \tag{1}$$

is minimized. $E_r(x_i, \mathcal{W})$ denotes the cost of assigning data point $x_i$ to code vector $w_r$.

Following an idea by (Luttrell, 1994) we consider the case that the code labels $r$ form a compressed encoding of the data for the purpose of transmission via a noisy channel (see Figure 1). The distortion caused by the channel noise is modeled by a matrix $H$ of transition probabilities $h_{rs}$, $\sum_s h_{rs} = 1$, $\forall r$, for the noise induced change of assignment of a data vector $x_i$ from code vector $w_r$ to code vector $w_s$. After transmission the received index $s$ is decoded using its code vector $w_s$. Averaging the squared Euclidean distance $\|x_i - w_s\|^2$ over all possible transitions yields the assignment costs

$$E_r(x_i, \mathcal{W}) = \frac{1}{2} \sum_s h_{rs} \|x_i - w_s\|^2 , \tag{2}$$

where the factor $1/2$ is introduced for computational convenience.

Starting from the cost function $E$ given in Eqs. (1), (2) the Gibbs-distribution $P(\mathcal{M}, \mathcal{W} \mid \mathcal{X}) = \frac{1}{Z} \exp(-\beta E(\mathcal{M}, \mathcal{W} \mid \mathcal{X}))$ can be obtained via the principle of maximum entropy under the constraint of a given average cost $\langle E \rangle$. The Lagrangian multiplier $\beta$ is associated with $\langle E \rangle$ and is interpreted as an inverse temperature that determines the fuzziness of assignments. In order to generalize from the given training set $\mathcal{X}$ we calculate the most likely set of code vectors from the probability distribution $P(\mathcal{M}, \mathcal{W} \mid \mathcal{X})$ marginalized over all legal sets of assignments $\mathcal{M}$. For a given value of $\beta$ we obtain

$$w_r = \frac{\sum_i x_i \sum_s h_{rs} P(x_i \in s)}{\sum_i \sum_s h_{rs} P(x_i \in s)} , \qquad \forall r , \tag{3}$$

where $P(x_i \in s) = \langle m_{is} \rangle$,

$$P(x_i \in s) = \frac{\exp\left(-\frac{\beta}{2} \sum_t h_{st} \|x_i - w_t\|^2\right)}{\sum_u \exp\left(-\frac{\beta}{2} \sum_t h_{ut} \|x_i - w_t\|^2\right)} , \tag{4}$$

is the assignment probability of data vector $x_i$ to code vector $w_s$. Solving Eqs. (3), (4) by fixed-point iteration comprises an expectation-maximization algorithm, where the E-step,

Figure 1: Cartoon of a generic data com-
munication problem. The encoder assigns
input vectors $x_i$ to labeled code vectors
$w_r$. Their indices r are then transmit-
ted via a noisy channel which is charac-
terized by a set of transition probabilities
$h_{rs}$. The decoder expands the received in-
dex s to its code vector $w_s$ which repre-
sents the data vectors assigned to it during
encoding. The total error is measured via
the squared Euclidean distance between
the original data vector $x_i$ and its repre-
sentative $w_s$ averaged over all transitions
$r \rightarrow s$.

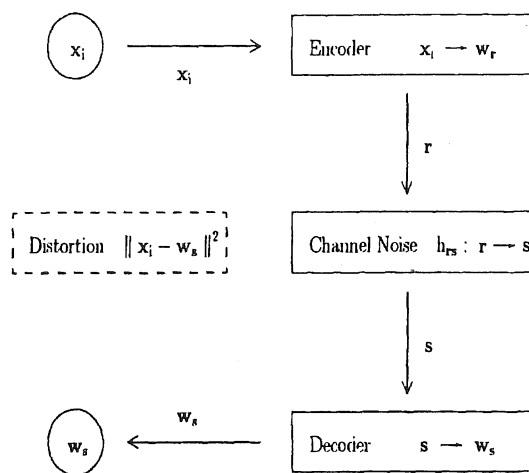

Eq. (4), determines the assignment probabilities $P(x_i \in s)$ for all data points $x_i$ and the
old code vectors $w_s$ and the M-step, Eq. (3), determines the new code vectors $w_r$ from the
new assignment probabilities $P(x_i \in s)$. In order to find the global minimum of E, $\beta = 0$
is increased according to an annealing schedule which tracks the solution from the easily
solvable convex problem at low $\beta$ to the exact solution of Eqs. (1), (2) at infinite $\beta$. In the
following we call the solution of Eqs. (3), (4) soft topographic vector quantizer (STVQ).

Eqs. (3), (4) are the starting point for a whole class of vector quantization algorithms (Fig-
ure 2). The approximation $h_{rs} \rightarrow \delta_{rs}$ applied to Eq. (4) leads to a soft version of Koho-
nen's self-organzing map (SSOM), if additionally applied to Eq. (3) soft-clustering (SC)
(Rose, 1990) is recovered. $\beta \rightarrow \infty$ leads to the corresponding "hard" versions topographic
vector quantisation (TVQ) (Luttrell, 1989), self-organizing map (SOM) (Kohonen, 1995),
and LBG. In the following, we will focus on the soft self-organizing map (SSOM). SSOM
is computationally less demanding than STVQ, but offers – in contrast to the traditional
SOM – a robust deterministic annealing optimization scheme. Hence it is possible to ex-
tend the SOM approach to arbitrary non-trivial neighborhood functions $h_{rs}$ as required,
e.g. for source channel coding problems for noisy channels.

# 3  PHASE TRANSITIONS IN THE ANNEALING

From (Rose, 1990) it is known that annealing in $\beta$ changes the representation of the data.
Code vectors split with increasing $\beta$ and the size of the codebook for a fixed $\beta$ is given by
the number of code vectors that have split up to that point. With non-diagonal H, however,
permutation symmetry is broken and the "splitting" behavior of the code vectors changes.

At infinite temperature every data vector $x_i$ is assigned to every code vector $w_r$ with equal
probability $P^0(x_i \in r) = 1/N$, where N is the size of the codebook. Hence all code
vectors are located in the center of mass, $w_r^0 = \frac{1}{D} \sum_i x_i$, $\forall r$, of the data. Expanding the
r.h.s. of Eq. (3) to first order around the fixed point $\{w_r^0\}$ and assuming $h_{rs} = h_{sr}$, $\forall r, s$,
we obtain the critical value

$$\beta^* = \frac{1}{\lambda_{max}^C \lambda_{max}^G} \tag{5}$$

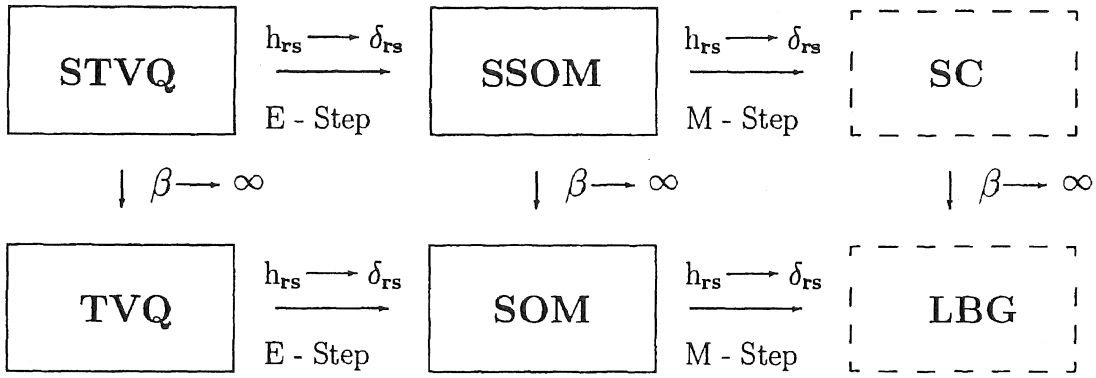

Figure 2: Class of vector quantizers derived from STVQ, together with approximations and limits (see text). The "S" in front stands for "soft" to indicate the probabilistic approach.

for the inverse temperature, at which the center of mass solution becomes unstable. $\lambda_{max}^C$ is the largest eigenvalue of the covariance matrix $C = \frac{1}{D}\sum_i x_i x_i^T$ of the data and corresponds to their variance $\lambda_{max}^C = \sigma_{max}^2$ along the principal axis which is given by the associated eigenvector $v_{max}^C$ and along which code vectors split. $\lambda_{max}^G$ is the largest eigenvalue of a matrix $G$ whose elements are given by $g_{rt} = \sum_s h_{rs}\left(h_{st} - \frac{1}{N}\right)$. The $r^{th}$ component of the corresponding eigenvector $v_{max}^G$ determines for each code vector $w_r$ in which direction along the principal axis it departs from $w_r^0$ and how it moves relative to the other code vectors. For SSOM a similar result is obtained with $G$ in Eq. (5) simply being replaced by $G^{SSOM}$, $g_{rt}^{SSOM} = h_{rt} - \frac{1}{N}$. See (Graepel, 1997) for details.

# 4 NUMERICAL RESULTS

In the following we consider a binary symmetric channel (BSC) with a bit error rate (BER) $\epsilon$. Assuming that the length of the code indices is n bits, the matrix elements of the transition matrix $H$ are

$$h_{rs} = \left(1 - \epsilon\right)^{n - d_H(r,s)} \epsilon^{d_H(r,s)}, \tag{6}$$

where $d_H(r, s)$ is the Hamming-distance between the binary representations of $r$ and $s$.

## 4.1 TOY PROBLEM

The numerical analysis of the phase transitions described in the previous section was performed on a toy data set consisting of 2000 data vectors drawn from a two-dimensional elongated Gaussian distribution $P(x) = (2\pi)^{-1}|C|^{-\frac{1}{2}}\exp(-\frac{1}{2}x^T C^{-1}x)$ with diagonal covariance matrix $C = \text{diag}(1, 0.04)$. The size of the codebook was $N = 4$ corresponding to $n = 2$ bits. Figure 3 (left) shows the x-coordinates of the positions of the code vectors in data space as functions of the inverse temperature $\beta$. At a critical inverse temperature $\beta^*$ the code vectors split along the x-axis which is the principal axis of the distribution of data points. In accordance with the eigenvector $v_{max}^G = (1, 0, 0, -1)^T$ for the largest eigenvalue $\lambda_{max}^G$ of the matrix $G$ two code vectors with Hamming distance $d_H = 2$ move to opposite positions along the principal axis, and two remain at the center. Note the degeneracy of eigenvalues for matrix (6). Figure 3 (right) shows the critical inverse temperature $\beta^*$ as a function of the BER for both STVQ (crosses) and SSOM (dots). Results are in very good agreement with the theoretical predictions of Eq. (5) (solid line). The inset displays the average cost $\langle E \rangle = \frac{1}{2}\sum_i \sum_r P(x_i \in r)\sum_s h_{rs}\|x_i - w_s\|^2$ as a function of $\beta$ for

$\epsilon = 0.08$ for STVQ and SSOM. The drop of the average cost occurs at the critical inverse temperature $\beta^*$.

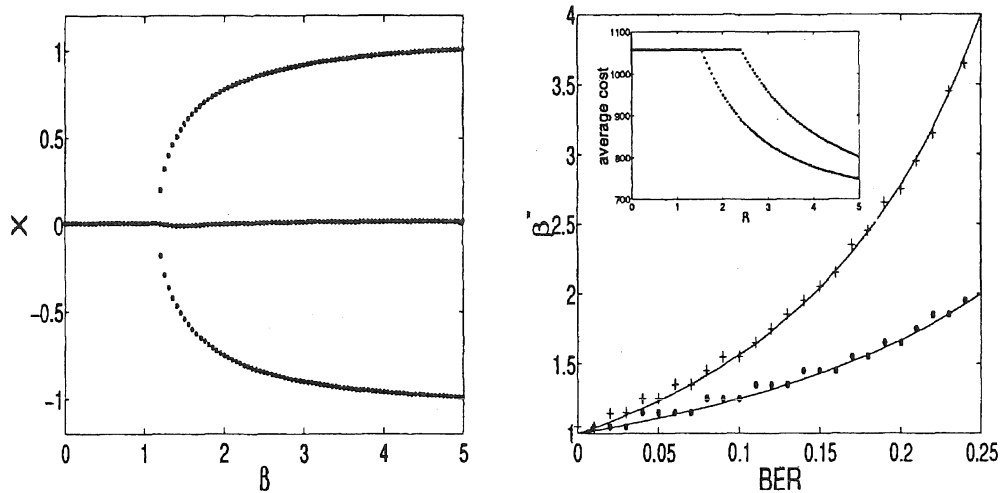

Figure 3: Phase transitions in the 2 bit "toy" problem. (left) X-coordinate of code vectors for the SSOM case plotted vs. inverse temperature $\beta$, $\epsilon = 0.08$. The splitting of the four code vectors occurs at $\beta = 1.25$ which is in very good accordance with the theory. (right) Critical values of $\beta$ for SSOM (dots) and STVQ (crosses), determined via the kink in the average cost (inset: $\epsilon = 0.08$, top line STVQ), which indicates the phase transition. Solid lines denote theoretical predictions. Convergence parameter for the fixed-point iteration, giving the upper limit for the difference in successive code vector positions per dimension, was $\delta = 5.0E - 10$.

## 4.2 SOURCE CHANNEL CODING FOR IMAGE DATA

In order to demonstrate the applicability of STVQ and in particular of SSOM to source channel coding we employed both algorithms to the compression of image data, which were then sent via a noisy channel and decoded after transmission. As a training set we used three $512 \times 512$ pixel 256 gray-value images from different scenes with blocksize $d = 2 \times 2$. The size of the codebook was chosen to be $N = 16$ in order to achieve a compression to 1 bpp. We applied an exponential annealing schedule given by $\beta_{t+1} = 2\beta_t$ and determined the start value $\beta_0$ to be just below the critical $\beta^*$ for the first split as given in Eq. (5). Note that with the transition matrix as given in Eq. (6) this optimization corresponds to the embedding of an $n = 4$ dimensional hypercube in the $d = 4$ dimensional data space. We tested the resulting codebooks by encoding our test image Lena[1] (Figure 5), which had not been used for determining the codebook, simulating the transmission of the indices via a noisy binary symmetric channel with given bit error rate and reconstructing the image using the codebook.

The results are summarized in Figure 4 which shows a plot of the signal-to-noise-ratio (SNR) as a function of the bit-error rate for STVQ (dots), SSOM (vertical crosses), and LBG (oblique crosses). STVQ shows the best performance especially for high BERs, where it is naturally far superior to the LBG-algorithm which does not take into account channel noise. SSOM, however, performs only slightly worse (approx. 1 dB) than STVQ. Considering the fact that SSOM is computationally much less demanding than STVQ

($\mathcal{O}(N)$ for encoding) - due to the omission of the convolution with $h_{rs}$ in Eq. (4) - the result demonstrates the efficiency of SSOM for source channel coding. Figure 4 also shows the generalization behavior of a SSOM codebook optimized for a BER of 0.05 (rectangles). Since this codebook was optimized for $\epsilon = 0.05$ it performs worse than appropriately trained SSOM codebooks for other values of BER, but still performs better than LBG except for low values of BERs. At low values, SSOMs trained for the noisy case are outperformed by LBG because robustness w.r.t. channel noise is achieved at the expense of an optimal data representation in the noise free case. Figure 5, finally, provides a visual impression of the performance of the different vector quantizers at a BER of 0.033. While the reconstruction for STVQ is only slightly better than the one for SSOM, both are clearly superior to the reconstruction for LBG.

Figure 4: Comparison between different vector quantizers for image compression, noisy channel (BSC) transmission and reconstruction. The plot shows the signal-to-noise-ratio (SNR), defined as $10\log_{10}(\sigma_{\text{signal}}/\sigma_{\text{noise}})$, as a function of bit-error rate (BER) for STVQ and SSOM, each optimized for the given channel noise, for SSOM, optimized for a BER of 0.05, and for LBG. The training set consisted of three $512 \times 512$ pixel 256 gray-value images with blocksize $d = 2 \times 2$. The codebook size was $N = 16$ corresponding to 1 bpp. The annealing schedule was given by $\beta_{t+1} = 2\,\beta_t$ and Lena was used as a test image. Convergence parameter $\delta$ was $1.0E - 5$.

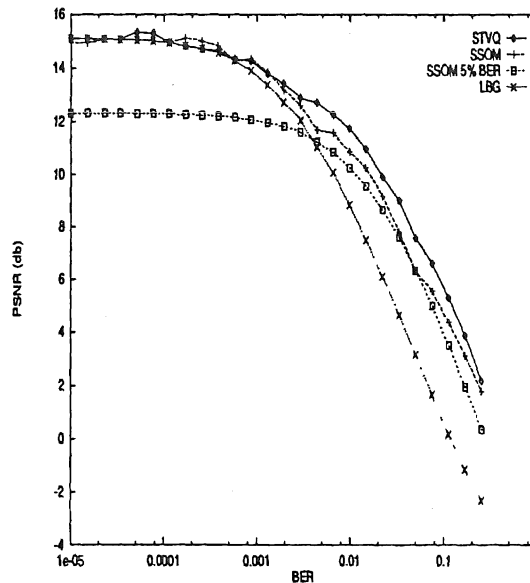

## 5 CONCLUSION

We presented an algorithm for noisy vector quantization which is based on deterministic annealing (STVQ). Phase transitions in the annealing process were analysed and a whole class of vector quantizers could be derived, includings standard algorithms such as LBG and "soft" versions as special cases of STVQ. In particular, a fuzzy version of Kohonen's SOM was introduced, which is computationally more efficient than STVQ and still yields very good results as demonstrated for noisy vector quantization of image data. The deterministic annealing scheme opens up many new possibilities for the usage of SOMs, in particular, when its neighborhood function represents non-trivial neighborhood relations.

**Acknowledgements** This work was supported by TU Berlin (FIP 13/41). We thank H. Bartsch for help and advice with regard to the image processing example.

## Footnotes

[1]The Lenna Story can be found at http://www.isr.com/ chuck/lennapg/lenna.shtml

## References

J. M. Buhmann and T. Hofmann. *Robust Vector Quantization by Competitive Learning.* Proceedings of ICASSP'97, Munich, (1997).

N. Farvardin. *A Study of Vector Quantization for Noisy Channels.* IEEE Transactions on Information Theory, vol. 36, p. 799-809 (1990).

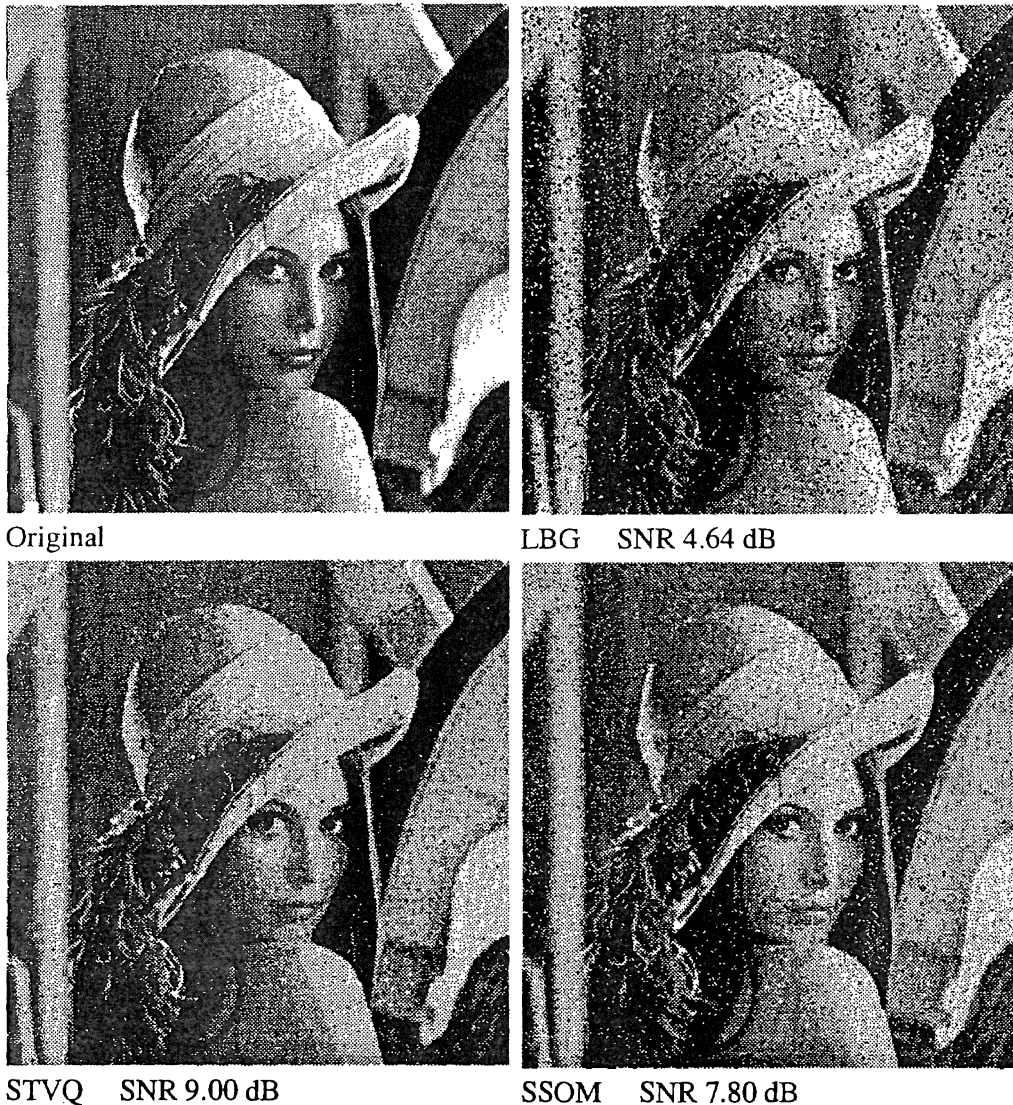

Original                                LBG      SNR 4.64 dB

STVQ      SNR 9.00 dB                   SSOM     SNR 7.80 dB

Figure 5: Lena transmitted over a binary symmetric channel with BER of 0.033 encoded and reconstructed using different vector quantization algorithms.

T. Graepel, M. Burger, and K. Obermayer. *Phase Transitions in Stochastic Self-Organizing Maps.* Physical Review E, vol. 56, no. 4, p. 3876-3890 (1997).

T. Heskes and B. Kappen. *Self-Organizing and Nonparametric Regression.* Artificial Neural Networks - ICANN'95, vol.1,p. 81-86 (1995).

T. Kohonen. *Self-Organizing Maps.* Springer-Verlag, 1995.

S. P. Luttrell. *Self-Organisation: A Derivation from first Principles of a Class of Learning Algorithms.* Proceedings of IJCNN'89, Washington DC, vol. 2, p. 495-498 (1989).

S. P. Luttrell. *A Baysian Analysis of Self-Organizing Maps.* Neural Computation, vol. 6, p. 767-794 (1994).

D. Miller and K. Rose. *Combined Source-Channel Vector Quantization Using Deterministic Annealing.* IEEE Transactions on Communications, vol. 42, p. 347-356 (1994).

K. Rose, E. Gurewitz, and G. C. Fox. *Statistical Mechanics and Phase Transitions in Clustering.* Physical Review Letters, vol. 65, No. 8, p. 945-948 (1990).

